# Unsupervised Color Constancy

**Kinh Tieu**
Artificial Intelligence Laboratory
Massachusetts Institute of Technology
Cambridge, MA 02139
`tieu@ai.mit.edu`

**Erik G. Miller**
Computer Science Division
UC Berkeley
Berkeley, CA 94720
`egmil@cs.berkeley.edu`

## Abstract

In [1] we introduced a linear statistical model of *joint color changes* in images due to variation in lighting and certain non-geometric camera parameters. We did this by measuring the mappings of colors in one image of a scene to colors in another image of the same scene under different lighting conditions. Here we increase the flexibility of this *color flow* model by allowing flow coefficients to vary according to a low order polynomial over the image. This allows us to better fit smoothly varying lighting conditions as well as curved surfaces without endowing our model with too much capacity. We show results on image matching and shadow removal and detection.

## 1 Introduction

The number of possible images of an object or scene, even when taken from a single viewpoint with a fixed camera, is very large. Light sources, shadows, camera aperture, exposure time, transducer non-linearities, and camera processing (such as auto-gain-control and color balancing) can all affect the final image of a scene. These effects have a significant impact on the images obtained with cameras and hence on image processing algorithms, often hampering or eliminating our ability to produce reliable recognition algorithms.

Addressing the variability of images due to these *photic parameters* has been an important problem in machine vision. We distinguish photic parameters from *geometric parameters*, such as camera orientation or blurring, that affect which parts of the scene a particular pixel represents. We also note that photic parameters are more general than "lighting parameters" and include anything which affects the final RGB values in an image given that the geometric parameters and the objects in the scene have been fixed.

We present a statistical linear model of *color change space* that is learned by observing how the colors in static images change *jointly* under common, naturally occurring lighting changes. Such a model can be used for a number of tasks, including synthesis of images of new objects under different lighting conditions, image matching, and shadow detection. Results for each of these tasks will be reported.

Several aspects of our model merit discussion. First, it is obtained from video data in a completely unsupervised fashion. The model uses no prior knowledge of lighting conditions, surface reflectances, or other parameters during data collection and modeling. It also has no built-in knowledge of the physics of image acquisition or "typical" image color

changes, such as brightness changes. Second, it is a single global model and does not need to be re-estimated for new objects or scenes. While it may not apply to all scenes equally well, it is a model of frequently occurring joint color changes, which is meant to apply to all scenes. Third, while our model is linear in *color change space*, each joint color change that we model (a 3-D vector field) is completely arbitrary, and is not itself restricted to being linear. This gives us great modeling power, while capacity is controlled through the number of basis fields allowed.

After discussing previous work in Section 2, we introduce the color flow model and how it is obtained from observations in Section 3. In Section 4, we show how the model and a single observed image can be used to generate a large family of related images. We also give an efficient procedure for finding the best fit of the model to the difference between two images. In Section 5 we give preliminary results for image matching (object recognition) and shadow detection.

## 2  Previous work

The color constancy literature contains a large body of work on estimating surface reflectances and various photic parameters from images. A common approach is to use linear models of reflectance and illuminant spectra [2]. Gray world algorithms [3] assume the average reflectance of all the surfaces in a scene is gray. White world algorithms [4] assume the brightest pixel corresponds to a scene point with maximal reflectance. Brainard and Freeman attacked this problem probabilistically [5] by defining prior distributions on particular illuminants and surfaces. They used a new, *maximum local mass* estimator to choose a single best estimate of the illuminant and surface.

Another technique is to estimate the relative illuminant or mapping of colors under an unknown illuminant to a canonical one. Color gamut mapping [6] uses the convex hull of all achievable RGB values to represent an illuminant. The intersection of the mappings for each pixel in an image is used to choose a "best" mapping. [7] trained a back-propagation multi-layer neural network to estimate the parameters of a linear color mapping. The approach in [8] works in the log color spectra space where the effect of a relative illuminant is a set of constant shifts in the scalar coefficients of linear models for the image colors and illuminant. The shifts are computed as differences between the modes of the distribution of coefficients of randomly selected pixels of some set of representative colors.

[9] bypasses the need to predict specific scene properties by proving that the set of images of a gray Lambertian convex object under all lighting conditions form a convex cone.[1] We wanted a model which, based upon a single image (instead of three required by [9]), could make useful predictions about other images of the same scene. This work is in the same spirit, although we use a statistical method rather than a geometric one.

## 3  Color flows

In the following, let $\mathcal{C} = \{(r, g, b)^T \in \mathbb{R}^3 : 0 \le r \le 255, 0 \le g \le 255, 0 \le b \le 255\}$ be the set of all possible observable image color 3-vectors. Let the vector-valued color of an image pixel $p$ be denoted by $\mathbf{c}(p) \in \mathcal{C}$.

Suppose we are given two $P$-pixel RGB color images $I_1$ and $I_2$ of the same scene taken under two different photic parameters $\theta_1$ and $\theta_2$ (the images are registered). Each pair of

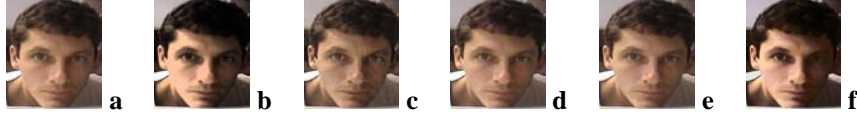

Figure 1: Matching non-linear color changes. **b** is the result of squaring the value of **a** (in HSV) and re-normalizing it to 255. **c-f** are attempts to match **b** with **a** using four different algorithms. Our algorithm (**f**) was the only one to capture the non-linearity.

corresponding image pixels $p_1^k$ and $p_2^k, 1 \le k \le P$, in the two images represents a single-color mapping $\mathbf{c}(p_1^k) \mapsto \mathbf{c}(p_2^k)$ that is conveniently represented by the vector difference:

$$\mathbf{d}(p_1^k, p_2^k) = \mathbf{c}(p_2^k) - \mathbf{c}(p_1^k). \tag{1}$$

By computing $P$ vector differences (one for each pair of pixels) and placing each at the point $\mathbf{c}(p_1^k)$ in color space $\mathcal{C}$, we have a *partially observed color flow*:

$$\Phi'(\mathbf{c}(p_1^k)) = \mathbf{d}(p_1^k, p_2^k), \qquad 1 \le k \le P \tag{2}$$

defined at points in $\mathcal{C}$ for which there are colors in image $I_1$.

To obtain a *full color flow* (i.e. a vector field $\Phi$ defined at all points in $\mathcal{C}$) from a partially observed color flow $\Phi'$, we must address two issues. First, there will be many points in $\mathcal{C}$ at which no vector difference is defined. Second, there may be multiple pixels of a particular color in image $I_1$ that are mapped to different colors in image $I_2$. We use a radial basis function estimator which defines the flow at a color point $(r, g, b)^T$ as the weighted proximity-based average of nearby observed "flow vectors". We found empirically that $\sigma^2 = 16$ (with colors on a 0–255 scale) worked well. Note that color flows are defined so that a color point with only a single nearby neighbor will inherit a flow vector that is nearly *parallel* to its neighbor. The idea is that if a particular color, under a photic parameter change $\theta_1 \mapsto \theta_2$, is observed to get a little bit darker and a little bit bluer, for example, then its neighbors in color space are also defined to exhibit this behavior.

## 3.1 Structure in the space of color flows

Consider a flat Lambertian surface that may have different reflectances as a function of the wavelength. While in principle it is possible for a change in lighting to map any color from such a surface to any other color *independently of all other colors*[2], we know from experience that many such joint maps are unlikely. This suggests that while the *marginal distribution* of mappings for a particular color is broadly distributed, the space of possible joint color maps (i.e., color flows) is much more compact[3].

In learning a statistical model of color flows, many common color flows can be anticipated such as ones that make colors a little darker, lighter, or more red. These types of flows can be well modeled with a simple global 3x3 matrix $\mathbf{A}$ that maps a color $\mathbf{c_1}$ in image $I_1$ to a color $\mathbf{c_2}$ in image $I_2$ via

$$\mathbf{c_2} = \mathbf{A}\mathbf{c_1}. \tag{3}$$

However, there are many effects which linear maps cannot model. Perhaps the most significant is the combination of a large brightness change coupled with a non-linear gain-control adjustment or brightness re-normalization by the camera. Such photic changes will tend

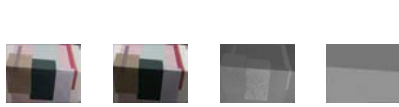

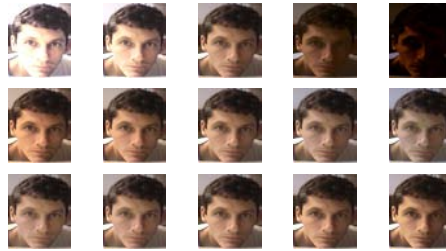

Figure 2: Evidence of non-linear color changes. The first two images are of the top and side of a box covered with multi-colored paper. The quotient image is shown next. The rightmost image is an ideal quotient image, corresponding to a linear lighting model.

Figure 3: Effects of the first three eigenflows. See text.

to leave the bright and dim parts of the image alone, while spreading the central colors of color space toward the margins.

For a linear imaging process, the ratio of the brightnesses of two images, or *quotient image* [12], should vary smoothly except at surface normal boundaries. However as shown in Figure 2, the quotient image is a function not only of surface normal, but also of albedo–direct evidence of a non-linear imaging process. Another pair of images exhibiting a non-linear color flow is shown in Figures 1**a** and **b**. Notice that the brighter areas of the original image get brighter and the darker portions get darker.

### 3.2 Color eigenflows

We wanted to capture the structure in color flow space by observing real-world data in an unsupervised fashion. A one square meter color palette was printed on standard non-glossy plotter paper using every color that could be produced by a Hewlett Packard DesignJet 650C. The poster was mounted on a wall in our office so that it was in the direct line of overhead lights and computer monitors but not the single office window. An inexpensive video camera (the PC-75WR, Supercircuits, Inc.) with auto-gain-control was aimed at the poster so that the poster occupied about 95% of the field of view.

Images of the poster were captured using the video camera under a wide variety of lighting conditions, including various intervals during sunrise, sunset, at midday, and with various combinations of office lights and outdoor lighting (controlled by adjusting blinds). People used the office during the acquisition process as well, thus affecting the ambient lighting conditions. It is important to note that a variety of non-linear normalization mechanisms built into the camera were operating during this process.

We chose image pairs $\mathcal{I}^j = (I_1^j, I_2^j), 1 \le j \le 800$, by randomly and independently selecting individual images from the set of raw images. Each image pair was then used to estimate a full color flow $\Phi(\mathcal{I}^j)$. We used 4096 distinct RGB colors (equally spaced in RGB space), so $\Phi(\mathcal{I}^j)$ was represented by a vector of $3 * 4096 = 12288$ components.

We modeled the space of color flows using principal components analysis (PCA) because: 1) the flows are well represented (in an $L_2$ sense) by a small number of principal components, and 2) finding the optimal description of a difference image in terms of color flows was computationally efficient using this representation (see Section 4). We call the principal components of the color flow data "color eigenflows", or just eigenflows,[4] for short. We emphasize that these principal components of color flows have *nothing to do with the distribution of colors in images*, but only model the distribution of *changes in color*. This is a key and potentially confusing point. Our work is very different from approaches that compute principal components in the intensity or color space itself [14, 15]. Perhaps the most important difference is that our model is a global model for all images, while the

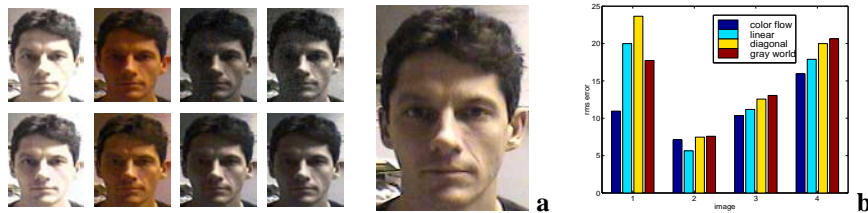

Figure 4: Image matching. Top row: original images. Bottom row: best approximation to original images using eigenflows and the source image **a.** Reconstruction errors per pixel component for four methods are shown in **b.**

above methods are models only for a particular set of images, such as faces.

## 4   Using color flows to synthesize novel images

How do we generate a new image from a source image and a color flow $\Phi$? For each pixel $p$ in the new image, its color $\mathbf{c}'(p)$ can be computed as

$$\mathbf{c}'(p) = \mathbf{c}(p) + \alpha\Phi(\hat{\mathbf{c}}(p)), \tag{4}$$

where $\mathbf{c}(p)$ is color in the source image and $\alpha$ is a scalar multiplier that represents the "quantity of flow". $\hat{\mathbf{c}}(p)$ is interpreted to be the color vector closest to $\mathbf{c}(p)$ (in color space) at which $\Phi$ has been computed. RGB values are clipped to 0–255.

Figure 3 shows the effect of the first three eigenflows on an image of a face. The original image is in the middle of each row while the other images show the application of each eigenflow with $\alpha$ values between $\pm 4$ standard deviations. The first eigenflow (top row) represents a generic brightness change that could probably be represented well with a linear model. Notice, however, the third row. Moving right from the middle image, the contrast grows. The shadowed side of the face grows darker while the lighted part of the face grows lighter. This effect cannot be achieved with a simple matrix multiplication as given in Equation 3. It is precisely these types of non-linear flows we wish to model.

We stress that the eigenflows were only computed once (on the color palette data), and that they were applied to the face image without any knowledge of the parameters under which the face image was taken.

### 4.1   Flowing one image to another

Suppose we have two images and we pose the question of whether they are images of the same object or scene. We suggest that if we can "flow" one image to another then the images are likely to be of the same scene.

Let us treat an image $I$ as a function that takes a color flow and returns a difference image $D$ by placing at each (x,y) pixel in $D$ the color change vector $\Phi(\mathbf{c}(p_{x,y}))$. The difference image basis for $I$ and set of eigenflows $\Psi_i, 1 \leq i \leq E$, is $D_i = I(\Psi_i)$. The set of images $\mathcal{S}$ that can be formed using a source image and a set of eigenflows is $\mathcal{S} = \{S : S = I + \sum_{i=1}^{E} \gamma_i D_i\}$, where the $\gamma_i$'s are scalars, and here $I$ is just an image, and not a function. In our experiments, we used $E = 30$ of the top eigenvectors.

We can only flow image $I_1$ to another image $I_2$ if it is possible to represent the difference image as a linear combination of the $D_i$'s, i.e. if $I_2 \in \mathcal{S}$. We find the optimal (in the least-squares sense) $\gamma_i$'s by solving the system

$$D = \sum_{i=1}^{E} \gamma_i D_i, \tag{5}$$

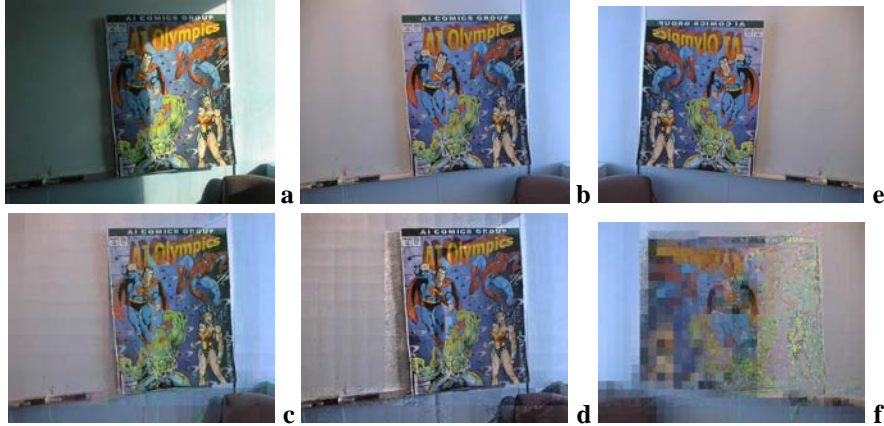

Figure 5: Modeling lighting changes with color flows. **a.** Image with strong shadow. **b.** Same image under more uniform lighting conditions. **c.** Flow from **a** to **b** using eigenflows. **d.** Flow from **a** to **b** using **linear**. Evaluating the capacity of the color flow model. **e.** Mirror image of **b**. **f.** Failure to flow **b** to **e** implies that the model is not overparameterized.

using the pseudo-inverse, where $D = I_2 - I_1$. The error residual represents a match score for $I_1$ and $I_2$. We point out again that this analysis ignores clipping effects. While clipping can only reduce the error between a synthetic image and a target image, it may change which solution is optimal in some cases.

## 5   Experiments

### 5.1   Image matching

One use of the color change model is for image matching. An ideal system would flow matching images with zero error, and have large errors for non-matching images.

We first examined our ability to flow a source image to a matching target image under different photic parameters. We compared our system to 3 other commonly used methods: **linear**, **diagonal**, and **gray world**. The **linear** method finds the matrix **A** in Equation 3 that minimizes the $L_2$ error between the synthetic and target images; **diagonal** does the same with a diagonal **A**; **gray world** linearly matches the mean R, G, B values of the synthetic and target images. While our goal was to reduce the numerical difference between two images using flows, it is instructive to examine one example that was particularly visually compelling, shown in Figure 1.

In a second experiment (Figure 4), we matched images of a face taken under various camera parameters but with constant lighting. Color flows outperforms the other methods in all but one task, on which it was second.

### 5.2   Local flows

In another test, the source and target images were taken under very different lighting conditions. Furthermore, shadowing effects and lighting direction changed between the two images. None of the methods could handle these effects when applied globally. Thus we repeatedly applied each method on small patches of the image. Our method again performed the best, with an RMS error of 13.8 per pixel component, compared with errors of 17.3, 20.1, and 20.6 for the other methods. Figure 5 shows obvious visual artifacts with the linear method, while our method seems to have produced a much better synthetic image, especially in the shadow region at the edge of the poster.

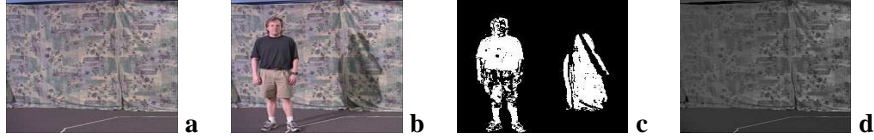

Figure 6: Backgrounding with color flows. **a** A background image. **b** A new object and shadow have appeared. **c** For each of the two regions (from background subtraction), a "flow" was done between the original image and the new image based on the pixels in each region. **d** The color flow of the original image using the eigenflow coefficients recovered from the shadow region. The color flow using the coefficients from the non-shadow region are unable to give a reasonable reconstruction of the new image.

Synthesis on patches of images greatly increases the capacity of the model. We performed one experiment to measure the over-fitting of our method versus the others by trying to flow an original image to its reflection (Figure 5). The RMS error per pixel component was $33.2$ for our method versus $41.5, 47.3$, and $48.7$ for the other methods. Note that while our method had lower error (which is undesirable), there was still a significant spread between matching images and non-matching images. We believe we can improve differentiation between matching and non-matching image pairs by assigning a cost to the *change* in $\gamma_i$ across each image patch. For non-matching images, we would expect the $\gamma_i$'s to vary rapidly to accommodate the changing image. For matching images, sharp changes would only be necessary at shadow boundaries or changes in the surface orientation relative to directional light sources.

## 5.3 Shadows

Shadows confuse tracking algorithms [16], backgrounding schemes and object recognition algorithms. For example, shadows can have a dramatic effect on the magnitude of difference images, despite the fact that no "new objects" have entered a scene. Shadows can also move across an image and appear as moving objects. Many of these problems could be eliminated if we could recognize that a particular region of an image is equivalent to a previously seen version of the scene, but under a different lighting.

Figure 6**a** shows how color flows may be used to distinguish between a new object and a shadow by flowing both regions. A constant color flow across an entire region may not model the image change well. However, we can extend our basic model to allow linearly or quadratically (or other low order polynomially) varying fields of eigenflow coefficients. That is, we can find the best least squares fit of the difference image allowing our $\gamma$ estimates to vary linearly or quadratically over the image. We implemented this technique by computing flows $\gamma_{x,y}$ between corresponding image patches (indexed by x and y), and then minimizing the following form:

$$\arg\min_{M} \sum_{x,y} (\gamma_{x,y} - Mc_{x,y})^T \Sigma_{x,y}^{-1} (\gamma_{x,y} - Mc_{x,y}). \qquad (6)$$

Here, each $c_{x,y}$ is a vector polynomial of the form $[x \ \ y \ \ 1]^T$ for the linear case and $[x^2 \ \ xy \ \ y^2 \ \ x \ \ y \ \ 1]^T$ for the quadratic case. $M$ is an $E$x3 matrix in the linear case and an $E$x6 matrix in the quadratic case. The $\Sigma_{x,y}^{-1}$'s are the error covariances in the estimate of the $\gamma_{x,y}$'s for each patch.

Allowing the $\gamma$'s to vary over the image greatly increases the capacity of a matcher, but by limiting this variation to linear or quadratic variation, the capacity is still not able to qualitatively match "non-matching" images. Note that this smooth variation in eigenflow coefficients can model either a nearby light source *or* a smoothly curving surface, since either of these conditions will result in a smoothly varying lighting change.

|            | constant | linear | quadratic |
|------------|----------|--------|-----------|
| shadow     | 36.5     | 12.5   | 12.0      |
| non-shadow | 110.6    | 64.8   | 59.8      |

Table 1: Error residuals for shadow and non-shadow regions after color flows.

We consider three versions of the experiment: 1) a single vector of flow coefficients, 2) linearly varying $\gamma$'s, 3) quadratically varying $\gamma$'s. In each case, the residual error for the shadow region is much lower than for the non-shadow region (Table 1).

### 5.4 Conclusions

Except for the synthesis experiments, most of the experiments in this paper are preliminary and only a proof of concept. Much larger experiments need to be performed to establish the utility of the color change model for particular applications. However, since the color change model represents a compact description of lighting changes, including non-linearities, we are optimistic about these applications.

## Footnotes

[1]This result depends upon the important assumption that the camera, including the transducers, the aperture, and the lens introduce no non-linearities into the system. The authors' results on color images also do not address the issue of metamers, and assume that light is composed of only the wavelengths red, green, and blue.

[2]By carefully choosing properties such as the surface reflectance of a point as a function of wavelength and lighting any mapping $\tilde{\Phi}$ can, in principle, be observed even on a flat Lambertian surface. However the metamerism which would cause such effects is uncommon in practice [10, 11]

[3]We will address below the significant issue of non-flat surfaces and shadows, which can cause highly 'incoherent' maps.

[4]PCA has been applied to *motion* vector fields [13], and these have also been termed "eigenflows".

## References

[1] E. Miller and K. Tieu. Color eigenflows: Statistical modeling of joint color changes. In *IEEE ICCV*, volume 1, pages 607–614, 2001.

[2] D. H. Marimont and B. A. Wandell. Linear models of surface and illuminant spectra. *J. Opt. Soc. Amer.*, 11, 1992.

[3] G. Buchsbaum. A spatial processor model for object color perception. *J. Franklin Inst.*, 310, 1980.

[4] J. J. McCann, J. A. Hall, and E. H. Land. Color mondrian experiments: The study of average spectral distributions. *J. Opt. Soc. Amer.*, A(67), 1977.

[5] D. H. Brainard and W. T. Freeman. Bayesian color constancy. *J. Opt. Soc. Amer.*, 14(7):1393–1411, 1997.

[6] D. A. Forsyth. A novel algorithm for color constancy. *IJCV*, 5(1), 1990.

[7] V. C. Cardei, B. V. Funt, and K. Barnard. Modeling color constancy with neural networks. In *Proc. Int. Conf. Vis., Recog., and Action: Neural Models of Mind and Machine*, 1997.

[8] R. Lenz and P. Meer. Illumination independent color image representation using log-eigenspectra. Technical Report LiTH-ISY-R-1947, Linköping University, April 1997.

[9] P. N. Belhumeur and D. Kriegman. What is the set of images of an object under all possible illumination conditions? *IJCV*, 28(3):1–16, 1998.

[10] W. S. Stiles, G. Wyszecki, and N. Ohta. Counting metameric object-color stimuli using frequency limited spectral reflectance functions. *J. Opt. Soc. Amer.*, 67(6), 1977.

[11] L. T. Maloney. Evaluation of linear models of surface spectral reflectance with small numbers of parameters. *J. Opt. Soc. Amer.*, A1, 1986.

[12] A. Shashua and R. Riklin-Raviv. The quotient image: Class-based re-rendering and recognition with varying illuminations. *IEEE PAMI*, 3(2):129–130, 2001.

[13] J. J. Lien. *Automatic Recognition of Facial Expressions Using Hidden Markov Models and Estimation of Expression Intensity*. PhD thesis, Carnegie Mellon University, 1998.

[14] M. Turk and A. Pentland. Eigenfaces for recognition. *J. Cog. Neuro.*, 3(1):71–86, 1991.

[15] M. Soriano, E. Marszalec, and M. Pietikainen. Color correction of face images under different illuminants by rgb eigenfaces. In *Proc. 2nd Int. Conf. on Audio- and Video-Based Biometric Person Authentication*, pages 148–153, 1999.

[16] K. Toyama, J. Krumm, B. Brumitt, and B. Meyers. Wallflower: Principles and practice of background maintenance. In *IEEE CVPR*, pages 255–261, 1999.
